# Visual Speech Recognition with Stochastic Networks

**Javier R. Movellan**
Department of Cognitive Science
University of California San Diego
La Jolla, Ca 92093-0515

## Abstract

This paper presents ongoing work on a speaker independent visual speech recognition system. The work presented here builds on previous research efforts in this area and explores the potential use of simple hidden Markov models for limited vocabulary, speaker independent visual speech recognition. The task at hand is recognition of the first four English digits, a task with possible applications in car-phone dialing. The images were modeled as mixtures of independent Gaussian distributions, and the temporal dependencies were captured with standard left-to-right hidden Markov models. The results indicate that simple hidden Markov models may be used to successfully recognize relatively unprocessed image sequences. The system achieved performance levels equivalent to untrained humans when asked to recognize the first four English digits.

## 1 INTRODUCTION

Visual articulation is an important source of information in face to face speech perception. Laboratory studies have shown that visual information allows subjects to tolerate an extra 4-dB of noise in the acoustic signal. This is particularly important considering that each decibel of signal to noise ratio translates into a 10-15% error reduction in the intelligibility of entire sentences (McCleod and Summerfield, 1990). Lip reading alone provides a basis for understanding for a large majority of the hearing impaired and when supplemented by acoustic or electrical signals it allows fluent understanding of speech in highly trained subjects. However visual information plays more than a simple compensatory role in speech perception. From early on humans are predisposed to integrate acoustic and visual information. Sensitivity to correspondences in auditory and visual information for speech events has been shown in 4 month old infants (Spelke, 1976; Kuhl & Meltzoff, 1982). By 6 years of age, humans consistently use audio visual contingencies to understand speech (Massaro, 1987). By adulthood, visual articulation automatically modulates perception of the acoustic signal. Under laboratory conditions it is possible to create powerful illusions in which subjects mistakenly hear sounds which are biased by visual articulations. Subjects in these experiments are typically unaware of

the discrepancy between the visual and auditory tracks and their experience is that of a unified auditory percept (McGurk & McDonnald, 1976).

Recent years have seen a revival of interest in audiovisual speech perception both in psychology and in the pattern recognition literature. There have been isolated efforts to build synthetic models of visual and audio-visual speech recognition (Petahan, 1985; Nishida, 1986; Yuhas, Goldstein, Sejnowski & Jenkins, 1988; Bregler, Manke, Hild & Waibel, 1993; Wolff, Prassad, Stork, & Hennecke, 1994). The main goal of these efforts has been to explore different architectures and visual processing techniques and to illustrate the potential use of visual information to improve the robustness of current speech recognition systems. Cognitive psychologists have also developed high level models of audio-visual speech perception that describe regularities in the way humans integrate visual and acoustic information (Massaro, 1987). In general these studies support the idea that human responses to visual and acoustic stimuli are conditional independent. This regularity has been used in some synthetic systems to simplify the task of integrating visual and acoustic signals (Wolff, Prassad, Stork, & Hennecke, 1994). Overall, multimodal speech perception is still an emerging field in which a lot of exploration needs to be done. The work presented here builds on the previous research efforts in this area and explores the potential use of simple hidden Markov models for limited vocabulary, speaker independent visual speech recognition. The task at hand is recognition of the first four English digits, a task with possible applications in car-phone dialing.

## 2  TRAINING SAMPLE

The training sample consisted of 96 digitized movies of 12 undergraduate students (9 males, 3 females) from the Cognitive Science Department at UCSD. Video capturing was performed in a windowless room at the Center for Research in Language at UCSD. Subjects were asked to talk into a video camera and to say the first four digits in English twice. Subjects could monitor the digitized images in a small display conveniently located in front of them. They were asked to position themselves so that that their lips be roughly centered in the feed-back display. Gray scale video images were digitized at 30 fps, 100x75 pixels, 8 bits per pixel. The video tracks were hand segmented by selecting a few relevant frames before and after the beginning and end of activity in the acoustic track. Statistics of the entire training sample are shown in table 1.

Table 1: Frame number statistics.

| Digit | Average | S.D. |
|-------|---------|------|
| "One"   | 8.9  | 2.1 |
| "Two"   | 9.6  | 2.1 |
| "Three" | 9.7  | 2.3 |
| "Four"  | 10.6 | 2.2 |

## 3  IMAGE PREPROCESSING

There are two different approaches to visual preprocessing in the visual speech recognition literature (Bregler, Manke, Hild & Waibel, 1993). The first approach, represented by the work of Wolff and colleagues (Wolff, Prassad, Stork, & Hennecke, 1994) favors sophisticated image preprocessing techniques to extract a limited set of hand-crafted features (e.g., height and width of the lips). The advantage of this approach is that it

drastically reduces the number of input dimensions. This translates into lower variability of the signal, potentially improved generalization, and large savings in computing time. The disadvantage is that vital information may be lost when compressing the image into a limited set of hand-crafted features. Variability is reduced at the possible expense of bias. Moreover, tests have shown that subtle holistic features such as the wrinkling and protrusion of the lips may play an important role in human lip-reading (Montgomery & Jackson, 1983). The second approach to visual preprocessing emphasizes preserving the original images as much as possible and letting the recognition engine discover the relevant features in the images. In most cases, images are low-pass filtered and dimension-reduced by using principal component analysis. The results in this papers indicate that good results can be obtained even without the use of principal components. In this investigation image preprocessing consisted of the following phases:

1. Symmetry enforcement: At each time frame the raw images were symmetrized by averaging pixel by pixel the left and right side of each image, using the vertical midline as the axis of symmetry. For convenience from now on we will refer to the raw images as "rho-images" and the symmetrized images as "sigma-images." The potential benefits of sigma-images are robustness, and compression, since the number of relevant pixels is reduced by half.

2. Temporal differentiaion: At each time frame we calculated the pixel by pixel differences between present sigma-images and immediately past sigma-images. For convenience we refer to the resulting images as "delta-images." One of the potential advantages of delta-images in the visual domain is their robustness to changes in illumination and the fact that they emphasize the dynamic aspects of the visual track.

3. Low pass filtering and subsampling: The sigma and delta images were compressed and subsampled using 20x15 equidistant Gaussian filters. Different values of the standard deviation of the Gaussian filters were tested.

4. Logistic thresholding and scaling: The sigma and delta images were independently thresholded by feeding the output of the Gaussian filters through a according to the following equation

$$y = 256 f(K \frac{\pi}{\sqrt{3}\sigma}(x - \mu))$$

where f is the logistic function, and $\mu, \sigma$, are respectively the average and standard deviation of the gray level distribution of entire image sequences. The constant K controls the sharpness of the logistic function. Assuming an approximately Gaussian distribution of gray levels when K=1 the thresholding function approximates histogram equalization, a standard technique in visual processing. Three different K values were tried: 0.3, 0.6 and 1.2.

5. Composites of the relevant portions of the blurred sigma and delta images were fed to the recognition network. The number of pixels of each processed image was 300 (150 from the blurred sigma images and 150 from the blurred delta images). Figure 1 shows the effect of the different preprocessing stages.

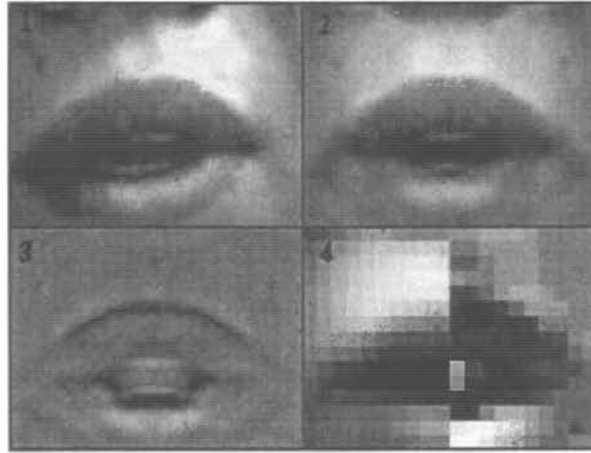

Figure 1: Image Preprocessing. 1) Rho-Image. 2) Sigma-Image.
3) Delta-Image. 4) Filtered and Sharpened Composite.

## 4  RECOGNITION NETWORK

We used the standard approach in limited vocabulary systems: a bank of hidden Markov models, one per word category, independently trained on the corresponding word categories. The images were modeled as mixtures of continuous probability distributions in pixel space. We tried mixtures of Gaussians and mixtures of Cauchy distributions. The mixtures of Cauchy distributions were very stable numerically but they did perform very poorly when compared to the Gaussian mixtures. We believe the reason for their poor performance is the tendency of Cauchy-based maximum-likelihood estimates to focus on individual exemplars. Gaussian-based estimates are much more prone to blend exemplars that belong to the same cluster. The initial state probabilities, transition probabilities, mixture coefficients, mixture centroids and variance parameters were trained using the E-M algorithm.

We initially encountered severe numerical underflow problems when using the E-M algorithm with Gaussian mixtures. These instabilities were due to the fact that the probability densities of images rapidly went to zero due to the large dimensionality of the images. Trimming the outputs of the Gaussian and using very small Gaussian gains did not work well. We solved the numerical problems in the following way: 1) Constraining all the variance parameters for all the states and mixtures to be equal. This allowed pulling out a constant in the likelihood-function of the mixtures, avoiding most numerical problems. 2) Initializing the mixture centroids using linear segmentation followed by the K-means clustering algorithm. For example, if there were 4 visual frames and 2 states, the first 2 frames were assigned to state 1 and the last 2 frames to state 2. K-means was then used independently on each of the states and their assigned frames. This is a standard initialization method in the acoustic domain (Rabiner & Bing-Hwang, 1993). Since K-means can be trapped in local minima, the algorithm was repeated 20 times with different starting point and the best solution was fed as the starting point for the E-M algorithm.

## 5  RESULTS

The main purpose of this study was to find simple image preprocessing techniques that would work well with hidden Markov models. We tested a wide variety of architectures and preprocessing parameters. In all cases the results were evaluated in terms of generalization to new speakers. Since the training sample is small, generalization performance was estimated using the jackknife procedure. Models were trained with 11

subjects, leaving one subject out for generalization testing. The entire procedure was repeated 12 times, each time leaving a different subject out for testing. Results are thus based on 96 generalization trials (4 digits x 12 subjects x 2 observations per subject). In all cases we tested several preprocessing techniques using 20 different architectures with different number of states (1,3,5,7,9) and mixtures per state (1,3,5,7). To compare the effect of each processing technique we used the average generalization performance of the best 4 architectures out of the 20 architectures tested.

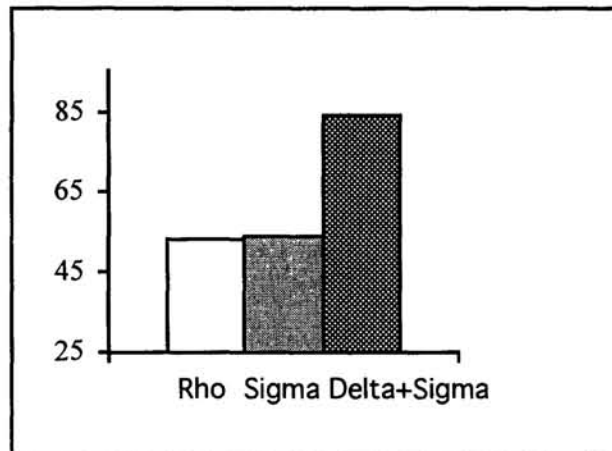

Figure 2: Average performance with the rho, sigma, and delta images.

Figure 2 shows the effects of symmetry enforcement and temporal differentiation. Symmetry enforcement had the benefit of reducing the input dimensionality by half and, as the figure show it did not hinder recognition performance. Using delta images had a very positive effect on recognition performance, as the figure shows. Figure 3 shows the effect of varying the thresholding constant and the standard deviation of the Gaussian filters. Best performance was obtained with blurring windows about 4 pixel wide and with thresholding just about histogram equalization.

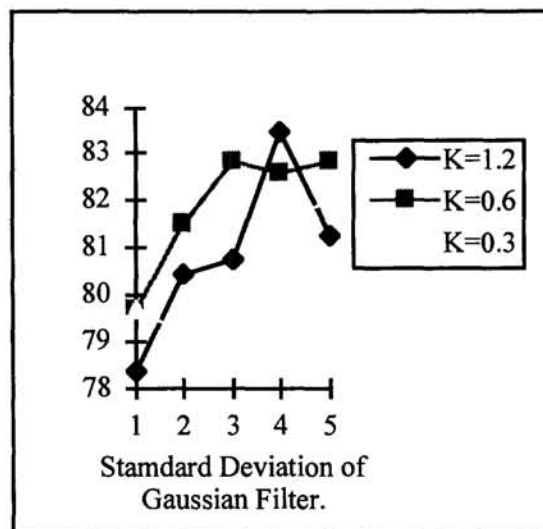

Figure 3: Effect of blurring and sharpening.

Table 2 shows the effects of variations in the number of states (S) and Gaussian mixtures (G) per state. The number within each cell is the percentage of simulations for which a particular combination of states and mixtures performed best out of the 20 architectures tested.

Table 2: Effect of varying the the number of states (S) and Gaussian mixtures (G).

|    | G1    | G3     | G5     | G7    |
|----|-------|--------|--------|-------|
| S1 | 0.00% | 0.00%  | 0.00%  | 0.00% |
| S3 | 0.00% | 21.87% | 12.5%  | 6.25% |
| S5 | 3.12% | 9.37%  | 15.62% | 0.00% |
| S7 | 6.25% | 12.5%  | 0.00%  | 3.12% |
| S9 | 6.25% | 0.00%  | 3.12%  | 0.00% |

Best overall performance was obtained with about 3 states and 3 mixtures per state. Peak performance was also obtained with a 3-state, 3-mixture per state network, with a generalization rate of 89.58% correct.

To compare these results with human performance, 9 subjects were tested on the same sample. Six subjects were normal hearing adults who were not trained in lip-reading. Three were hearing impaired with profound hearing loss and had received training in lip reading at 2 to 8 years of age. The mean correct response for normal subjects was 89.93 % correct, just about the same rate as the best artificial network. The hearing impaired had an average performance of 95.49% correct, significantly better than our network.

Table 3: Confusion matrix of the best artificial system.

|         | 1       | 2      | 3      | 4      |
|---------|---------|--------|--------|--------|
| "One"   | 100.00% | 0.00%  | 0.00%  | 0.00%  |
| "Two"   | 4.17%   | 87.50% | 4.17%  | 4.17%  |
| "Three" | 12.5%   | 0.00%  | 83.33% | 4.17%  |
| "Four"  | 8.33%   | 4.17%  | 0.00%  | 87.50% |

Table 4: Average human confusion matrix.

|         | 1      | 2      | 3      | 4      |
|---------|--------|--------|--------|--------|
| "One"   | 89.36% | 0.46%  | 8.33%  | 1.85%  |
| "Two"   | 1.39%  | 98.61% | 0.00%  | 0.00%  |
| "Three" | 9.25%  | 3.24%  | 85.64% | 1.87%  |
| "Four"  | 4.17%  | 0.46%  | 1.85%  | 93.52% |

Tables 3 and 4 show the confusion matrices for the best network and the average confusion matrix with all 9 subjects combined. The correlation between these two matrices was 0.99. This means that 98% of the variance in human confusions can be accounted for by the artificial model. This suggests that the representational space learned by the artificial system may be a reasonable model of the representational space used by humans. Figure 5 shows the representations learned by a network with 6 states and 1 mixture per state. Each column is a different digit, starting with "one." Each row

is a different temporal state. The two pictures within each cell are sigma and delta image centroids. As the figure shows, the identity of individual exemplars is lost but the underlying dynamics of the digits are preserved. The digits can be easily recognized when played as a movie.

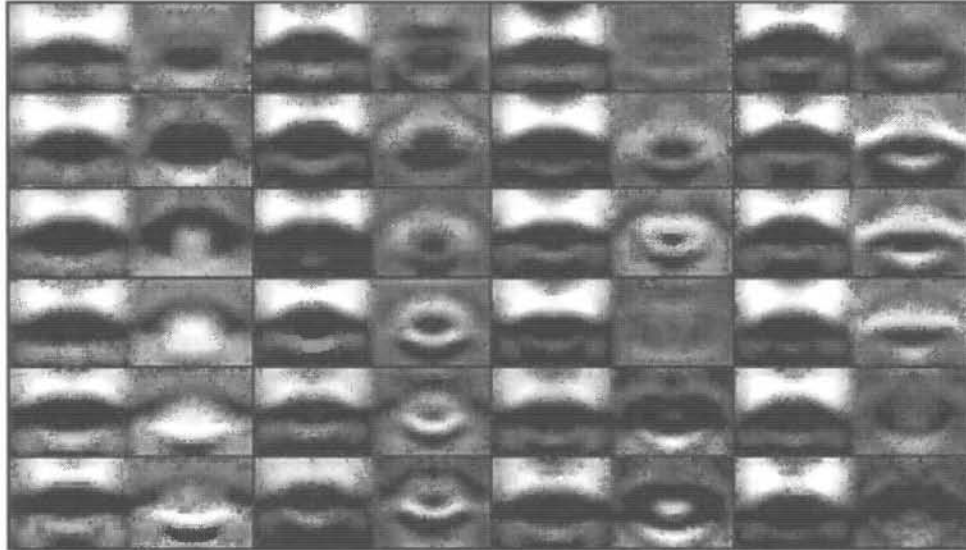

Figure 4: Dynamic representations learned by a simple network.

## 6 CONCLUSIONS

This paper shows that simple stochastic networks, like hidden Markov models, can be successfully applied for visual speech recognition using relatively unprocessed images. The performance level obtained with these networks roughly matches untrained human performance. Moreover, the representational space learned by these networks may be a reasonable model of the representations used by humans. More research should be done to better understand how humans integrate visual and acoustic information in speech perception and to develop practical models for robust audio-visual speech recognition.

## References

Bregler C., Manke S., Hild H. & Waibel A. (1993) Bimodal Sensor Integration on the Example of "Speech-Reading". *Proc ICNN-93*, II,667-677.

Kuhl P. & Meltzoff A (1982) The Bimodal Development of Speech in Infancy. *Science*, 218, 1138-1141.

MacLeod A. & Summerfield Q. (1990) A Procedure for Measuring Auditory and Audio-visual Speech-Reception Measuring Thresholds for Sentences in Noise: Rationale, Evaluation and Recommendations for Use. *British Journal of Audiology*. 24, 29-43.

Massaro D. (1987) Speech Perception by Ear and Eye. In Dodd B. & Campbell R. (ed.) *Hearing by Eye: The Psychology of Lip-Reading*. London, LEA, 53-83.

Massaro D., Cohen M & Getsi (1993) Long-Term Training, Transfer and Retention in Learning to Lip-read. *Perception and Psychophysics,* 53, 549-562.

McGurk H. & MacDonald J. (1976) Hearing Lips and Seeing Voices. *Nature*, 264, 126-130.

Montgomery A. & Jackson P. (1983) Physical Characteristics of the Lips Underlying Vowel Lipreading Performance. *Journal of the Acoustical Society of America*, 73,2134-2144.

Nishida S. (1986) Speech Recognition Enhancement by Lip Information. *Proceedings of ACM/CHI 86*, 198-204.

Petajan E. (1985) Automatic Lip Reading to Enhance Speech Recognition. *IEEE CVPR 85*, 40-47.

Rabiner L., Bing-Hwang J. (1993) *Fundamentals of Speech Recognition*. New Jersey, Prentice Hall.

Spelke E. (1976) Infant's Intermodal Perception of Events. *Cognitive Psychology*, 8, 533-560.

Yuhas B., Goldstein T., Sejnowski T., Jenkins R. (1988) Neural Network Models of Sensory Integration for Improved Vowel Recognition. *Proceedings IEEE 78*, 1655-1668.

Wolff G., Prassad L., Stork D., Hennecke M. (1994) Lipreading by Neural Networks: Visual Preprocessing, Learning and Sensory Integration. In J. Cowan, G. Tesauro, J. Alspector (ed.), *Advances in Neural Information Processing Systems 6*, 1027-1035. San Mateo, CA: Morgan Kaufmann.
